# Correlates of Attention in a Model of Dynamic Visual Recognition*

Rajesh P. N. Rao
Department of Computer Science
University of Rochester
Rochester, NY 14627
rao@cs.rochester.edu

## Abstract

Given a set of objects in the visual field, how does the the visual system learn to attend to a particular object of interest while ignoring the rest? How are occlusions and background clutter so effortlessly discounted for when recognizing a familiar object? In this paper, we attempt to answer these questions in the context of a Kalman filter-based model of visual recognition that has previously proved useful in explaining certain neurophysiological phenomena such as endstopping and related extra-classical receptive field effects in the visual cortex. By using results from the field of robust statistics, we describe an extension of the Kalman filter model that can handle multiple objects in the visual field. The resulting robust Kalman filter model demonstrates how certain forms of attention can be viewed as an emergent property of the interaction between top-down expectations and bottom-up signals. The model also suggests functional interpretations of certain attention-related effects that have been observed in visual cortical neurons. Experimental results are provided to help demonstrate the ability of the model to perform robust segmentation and recognition of objects and image sequences in the presence of varying degrees of occlusions and clutter.

## 1 INTRODUCTION

The human visual system possesses the remarkable ability to recognize objects despite the presence of distractors and occluders in the field of view. A popular suggestion is that an "attentional spotlight" mediates this ability to preferentially process a relevant object in a given scene (see [5, 9] for reviews). Numerous models have been proposed to simulate the control of this "focus of attention" [10, 11, 15]. Unfortunately, there is inconclusive evidence for the existence of an explicit neural mechanism for implementing an attentional spotlight in the visual

cortex. Thus, an important question is whether there are alternate neural mechanisms which don't explicitly use a spotlight but whose effects can nevertheless be interpreted as attention. In other words, can attention be viewed as an emergent property of a distributed network of neurons whose primary goal is visual recognition?

In this paper, we extend a previously proposed Kalman filter-based model of visual recognition [13, 12] to handle the case of multiple objects, occlusions, and clutter in the visual field. We provide simulation results suggesting that certain forms of attention can be viewed as an emergent property of the interaction between bottom-up signals and top-down expectations during visual recognition. The simulation results demonstrate how "attention" can be switched between different objects in a visual scene without using an explicit spotlight of attention.

## 2  A KALMAN FILTER MODEL OF VISUAL RECOGNITION

We have previously introduced a hierarchical Kalman filter-based model of visual recognition and have shown how this model can be used to explain neurophysiological effects such as end-stopping and neural response suppression during free-viewing of natural images [12, 13]. The Kalman filter [7] is essentially a linear dynamical system that attempts to mimic the behavior of an observed natural process. At any time instant $t$, the filter assumes that the internal state of the given natural process can be represented as a $k \times 1$ vector $\mathbf{r}(t)$. Although not directly accessible, this internal state vector is assumed to generate an $n \times 1$ measurable and observable output vector $\mathbf{I}(t)$ (for example, an image) according to:

$$\mathbf{I}(t) = U\mathbf{r}(t) + \mathbf{n}(t) \tag{1}$$

where $U$ is an $n \times k$ *generative (or measurement) matrix*, and $\mathbf{n}(t)$ is a Gaussian stochastic noise process with mean zero and a covariance matrix given by $\Sigma = E[\mathbf{n}\mathbf{n}^T]$ ($E$ denotes the expectation operator and $T$ denotes transpose).

In order to specify how the internal state $\mathbf{r}$ changes with time, the Kalman filter assumes that the process of interest can be modeled as a *Gauss-Markov random process* [1]. Thus, given the state $\mathbf{r}(t-1)$ at time instant $t-1$, the next state $\mathbf{r}(t)$ is given by:

$$\mathbf{r}(t) = V\mathbf{r}(t-1) + \mathbf{m}(t-1) \tag{2}$$

where $V$ is the *state transition (or prediction) matrix* and $\mathbf{m}$ is white Gaussian noise with mean $\overline{\mathbf{m}} = E[\mathbf{m}]$ and covariance $\Pi = E[(\mathbf{m} - \overline{\mathbf{m}})(\mathbf{m} - \overline{\mathbf{m}})^T]$.

Given the generative model in Equation 1 and the dynamics in Equation 2, the goal is to optimally estimate the current internal state $\mathbf{r}(t)$ using only the measurable inputs $\mathbf{I}(t)$. An optimization function whose minimization yields an estimate of $\mathbf{r}$ is the *weighted least-squares criterion*:

$$J = (\mathbf{I} - U\mathbf{r})^T \Sigma^{-1} (\mathbf{I} - U\mathbf{r}) + (\mathbf{r} - \overline{\mathbf{r}})^T M^{-1} (\mathbf{r} - \overline{\mathbf{r}}) \tag{3}$$

where $\overline{\mathbf{r}}(t)$ is the mean of the state vector *before* measurement of the input data $\mathbf{I}(t)$ and $M = E[(\mathbf{r} - \overline{\mathbf{r}})(\mathbf{r} - \overline{\mathbf{r}})^T]$ is the corresponding covariance matrix. It is easy to show [1] that $J$ is simply the sum of the negative log-likelihood of generating the data $\mathbf{I}$ given the state $\mathbf{r}$, and the negative log of the prior probability of the state $\mathbf{r}$. Thus, minimizing $J$ is equivalent to maximizing the posterior probability $p(\mathbf{r}|\mathbf{I})$ of the state $\mathbf{r}$ given the input data.

The optimization function $J$ can be minimized by setting $\frac{\partial J}{\partial \mathbf{r}} = 0$ and solving for the minimum value $\hat{\mathbf{r}}$ of the state $\mathbf{r}$ (note that $\hat{\mathbf{r}}$ equals the mean of $\mathbf{r}$ after measurement of $\mathbf{I}$). The resultant *Kalman filter* equation is given by:

$$\hat{\mathbf{r}}(t) = \overline{\mathbf{r}}(t) + N(t)U^T \Sigma(t)^{-1} (\mathbf{I}(t) - U\overline{\mathbf{r}}(t)) \tag{4}$$

$$\overline{\mathbf{r}}(t) = V\hat{\mathbf{r}}(t-1) + \overline{\mathbf{m}}(t-1) \tag{5}$$

where $N(t) = (U^T \Sigma(t)^{-1} U + M(t)^{-1})^{-1}$ is a "normalization" matrix that maintains the covariance of the state $\mathbf{r}$ after measurement of $\mathbf{I}$. The matrix $M$, which is the covariance before

measurement of $\mathbf{I}$, is updated as $M(t) = VN(t-1)V^T + \Pi(t-1)$. Thus, the Kalman filter predicts one step into the future using Equation 5, obtains the next sensory input $\mathbf{I}(t)$, and then corrects its prediction $\bar{\mathbf{r}}(t)$ using the sensory residual error $(\mathbf{I}(t) - U\bar{\mathbf{r}}(t))$ and the Kalman gain $N(t)U^T\Sigma(t)^{-1}$. This yields the corrected estimate $\hat{\mathbf{r}}(t)$ (Equation 4), which is then used to make the next state prediction $\bar{\mathbf{r}}(t+1)$.

The measurement (or generative) matrix $U$ and the state transition (or prediction) matrix $V$ used by the Kalman filter together encode an *internal model* of the observed dynamic process. As suggested in [13], it is possible to *learn* an internal model of the input dynamics from observed data. Let $\mathbf{u}$ and $\mathbf{v}$ denote the vectorized forms of the matrices $U$ and $V$ respectively. For example, the $n \times k$ generative matrix $U$ can be collapsed into an $nk \times 1$ vector $\mathbf{u} = [U^1 U^2 \ldots U^n]^T$ where $U^i$ denotes the $i$th row of $U$. Note that $(\mathbf{I} - U\mathbf{r}) = (\mathbf{I} - R\mathbf{u})$ where $R$ is the $n \times nk$ matrix given by:

$$R = \begin{bmatrix} \mathbf{r}^T & 0 & \ldots & 0 \\ 0 & \mathbf{r}^T & \ldots & 0 \\ \vdots & \vdots & \vdots & \vdots \\ 0 & \ldots & 0 & \mathbf{r}^T \end{bmatrix} \qquad (6)$$

By minimizing an optimization function similar to $J$ [13], one can derive a Kalman filter-like "learning rule" for the generative matrix $U$:

$$\hat{\mathbf{u}}(t) = \bar{\mathbf{u}}(t) + N_u(t)R(t)^T\Sigma(t)^{-1}(\mathbf{I}(t) - R(t)\bar{\mathbf{u}}(t)) - \alpha N_u(t)\bar{\mathbf{u}}(t) \qquad (7)$$

where $\bar{\mathbf{u}}(t) = \hat{\mathbf{u}}(t-1)$, $N_u(t) = (N_u(t-1)^{-1} + R(t)^T\Sigma(t)^{-1}R(t) + \alpha I)^{-1}$, and $I$ is the $nk \times nk$ identity matrix. The constant $\alpha$ determines the decay rate of $\bar{\mathbf{u}}$.

As in the case of $U$, an estimate of the prediction matrix $V$ can be obtained via the following learning rule for $\mathbf{v}$ [13]:

$$\hat{\mathbf{v}}(t) = \bar{\mathbf{v}}(t) + N_v(t)\hat{R}(t)^T M(t)^{-1}[\mathbf{r}(t+1) - \bar{\mathbf{r}}(t+1)] - \beta N_v(t)\bar{\mathbf{v}}(t) \qquad (8)$$

where $\bar{\mathbf{v}}(t) = \hat{\mathbf{v}}(t-1)$, $N_v(t) = (N_v(t-1)^{-1} + \hat{R}(t)^T M(t)^{-1}\hat{R}(t) + \beta I)^{-1}$ and $\hat{R}$ is a $k \times k^2$ matrix analogous to $R$ (Equation 6) but with $\mathbf{r}^T = \hat{\mathbf{r}}^T$. The constant $\beta$ determines the decay rate for $\mathbf{v}$ while $I$ denotes the $k^2 \times k^2$ identity matrix. Note that in this case, the estimate of $V$ is corrected using the prediction residual error $(\mathbf{r}(t+1) - \bar{\mathbf{r}}(t+1))$, which denotes the difference between the actual state and the predicted state. One unresolved issue is the specification of values for $\mathbf{r}(t)$ (comprising $R(t)$) in Equation 7 and $\mathbf{r}(t+1)$ in Equation 8. The Expectation-Maximization (EM) algorithm [4] suggests that in the case of static stimuli $(\bar{\mathbf{r}}(t) = \hat{\mathbf{r}}(t-1))$, one may use $\mathbf{r}(t) = \hat{\mathbf{r}}$ which is the converged optimal state estimate for the given static input. In the case of dynamic stimuli, the EM algorithm prescribes $\mathbf{r}(t) = \hat{\mathbf{r}}(t|N)$, which is the optimal temporally *smoothed* state estimate [1] for time $t$ ($\leq N$), given input data for each of the time instants $1, \ldots, N$. Unfortunately, the smoothed estimate requires knowledge of future inputs and is computationally quite expensive. For the experimental results, we used the on-line estimates $\hat{\mathbf{r}}(t)$ when updating the matrices $U$ and $V$ during training.

## 3  ROBUST KALMAN FILTERING

The standard derivation of the Kalman filter minimizes Equation 3 but unfortunately does not specify how the covariance $\Sigma$ is to be obtained. A common choice is to use a constant matrix or even a constant scalar. Making $\Sigma$ constant however reduces the Kalman filter estimates to standard least-squares estimates, which are highly susceptible to outliers or gross errors i.e. data points that lie far away from the bulk of the observed or predicted data [6]. For example, in the case where $\mathbf{I}$ represents an input image, occlusions and clutter will cause many pixels in $\mathbf{I}$ to deviate significantly from corresponding pixels in the predicted image $U\mathbf{r}$. The problem

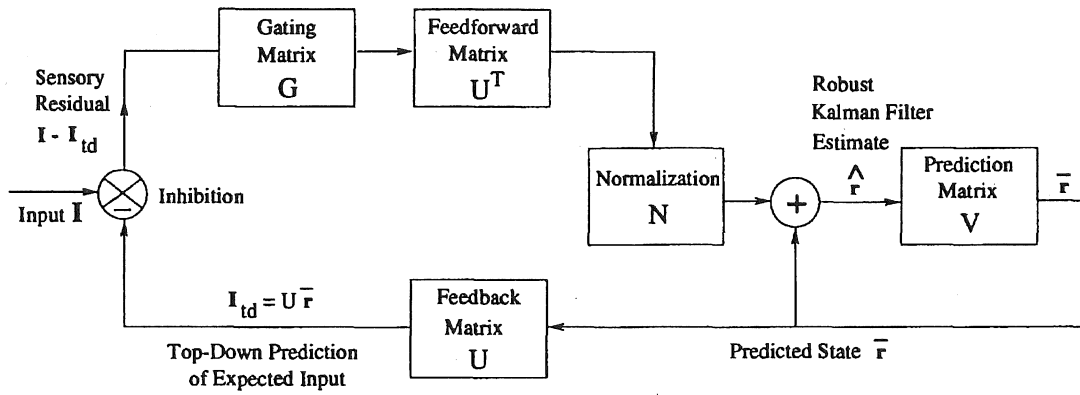

Figure 1: **Recurrent Network Implementation of the Robust Kalman Filter**. The gating matrix $G$ is a non-linear function of the current residual error between the input $\mathbf{I}$ and its top-down prediction $U\bar{\mathbf{r}}$. $G$ effectively filters out any high residuals, thereby preventing outliers in input data $\mathbf{I}$ from influencing the robust Kalman filter estimate $\hat{\mathbf{r}}$. Note that the entire filter can be implemented in a recurrent neural network, with $U$, $U^T$, and $V$ represented by the synaptic weights of neurons with linear activation functions and $G$ being implemented by a set of threshold non-linear neurons with binary outputs.

of outliers can be tackled using *robust estimation procedures* [6] such as M-estimation, which involves minimizing a function of the form:

$$J' = \sum_{i=1}^{n} \rho \left( \mathbf{I}^i - U^i \mathbf{r} \right) \tag{9}$$

where $\mathbf{I}^i$ and $U^i$ are the $i$th pixel and $i$th row of $\mathbf{I}$ and $U$ respectively, and $\rho$ is a function that increases less rapidly than the square. This reduces the influence of large residual errors (which correspond to outliers) on the optimization of $J'$, thereby "rejecting" the outliers. A special case of the above function is the following weighted least squares criterion:

$$J' = (\mathbf{I} - U\mathbf{r})^T S (\mathbf{I} - U\mathbf{r}) \tag{10}$$

where $S$ is a diagonal matrix whose diagonal entries $S^{i,i}$ determine the weight accorded to the corresponding pixel error $(\mathbf{I}^i - U^i \mathbf{r})$. A simple but attractive choice for these weights is the non-linear function given by $S^{i,i} = \min\left\{1, c/(\mathbf{I}^i - U^i \mathbf{r})^2\right\}$, where $c$ is a threshold parameter. To understand the behavior of this function, note that $S$ effectively clips the $i$th summand in $J'$ (Equation 10 above) to a constant value $c$ whenever the $i$th squared residual $(\mathbf{I}^i - U^i \mathbf{r})^2$ exceeds the threshold $c$; otherwise, the summand is set equal to the squared residual.

By substituting $\Sigma^{-1} = S$ in the optimization function $J$ (Equation 3), we can rederive the following *robust Kalman filter* equation:

$$\hat{\mathbf{r}}(t) = \bar{\mathbf{r}}(t) + N(t) U^T G(t) (\mathbf{I} - U\bar{\mathbf{r}}(t)) \tag{11}$$

where $\bar{\mathbf{r}}(t) = V\hat{\mathbf{r}}(t-1)) + \overline{\mathbf{m}}(t-1)$, $N(t) = (U^T G(t) U + M(t)^{-1})^{-1}$, $M(t) = VN(t-1)V^T + \Pi(t-1)$, and $G(t)$ is an $n \times n$ diagonal matrix whose diagonal entries at time instant $t$ are given by:

$$G^{i,i} = \begin{cases} 0 & \text{if } (\mathbf{I}^i(t) - U^i \bar{\mathbf{r}}(t))^2 > c(t) \\ 1 & \text{otherwise} \end{cases}$$

$G$ can be regarded as the sensory residual gain or "gating" matrix, which determines the (binary) gain on the various components of the incoming sensory residual error vector. By effectively filtering out any high residuals, $G$ allows the Kalman filter to ignore the corresponding outliers in the input $\mathbf{I}$, thereby enabling it to robustly estimate the state $\mathbf{r}$. Figure 1 depicts an implementation of the robust Kalman filter in the form of a recurrent network of linear and threshold non-linear neurons. In particular, the feedforward, feedback and prediction neurons possess linear activation functions while the gating neurons implementing $G$ compute binary outputs based on a threshold non-linearity.

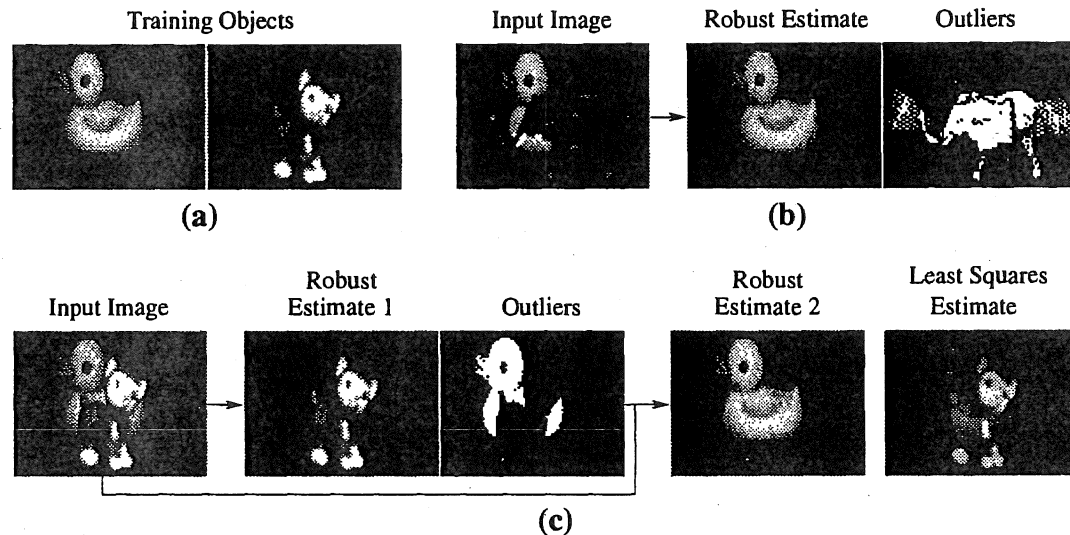

Figure 2: **Correlates of Attention during Static Recognition.** (a) Images of size $105 \times 65$ used to train a robust Kalman filter network. The generative matrix $U$ was $6825 \times 5$. (b) Occlusions and background clutter are treated as outliers (white regions in the third image, depicting the diagonal of the gating matrix $G$). This allows the network to "attend to" and recognize the training object, as indicated by the accurate reconstruction (middle image) of the training image based on the final robust state estimate. (c) In the more interesting case of the training objects occluding each other, the network converges to one of the objects (the "dominant" one in the image - in this case, the object in the foreground). Having recognized one object, the second object is attended to and recognized by taking the complement of the outliers (diagonal of $G$) and repeating the robust filtering process (third and fourth images). The fifth image is the image reconstruction obtained from the standard (least squares derived) Kalman filter estimate, showing an inability to resolve or recognize either of the two objects.

## 4 VISUAL ATTENTION IN A SIMULATED NETWORK

The gating matrix $G$ allows the Kalman filter network to "selectively attend" to an object while treating the remaining components of the sensory input as outliers. We demonstrate this capability of the network using three different examples. In the first example, a network was trained on static grayscale images of a pair of $3D$ objects (Figure 2 (a)). For learning static inputs, the prediction matrix $V$ is unnecessary since we may use $\bar{r}(t) = \hat{r}(t-1)$ and $M(t) = N(t-1)$. After training, the network was tested on images containing the training objects with varying degrees of occlusion and clutter (Figure 2 (b) and (c)). The outlier threshold $c$ was initialized to the sum of the mean plus $k$ standard deviations of the current distribution of squared residual errors $(I^i - U^i r)^2$. The value of $k$ was gradually decreased during each iteration in order to allow the network to refine its robust estimate by gradually pruning away the outliers as it converges to a single object estimate. After convergence, the diagonal of the matrix $G$ contains zeros in the image locations containing the outliers and ones in the remaining locations. As shown in Figure 2 (b), the network was successful in recognizing the training object despite occlusion and background clutter.

More interestingly, the outliers (white) produce a crude *segmentation* of the occluder and background clutter, which can subsequently be used to focus "attention" on these previously ignored objects and recover their identity. In particular, an *outlier mask* m can be defined by taking the complement of the diagonal of $G$ (i.e. $m^i = 1 - G^{i,i}$). By replacing the diagonal of $G$ with m in Equation 11[1] and repeating the estimation process, the network can "attend to"

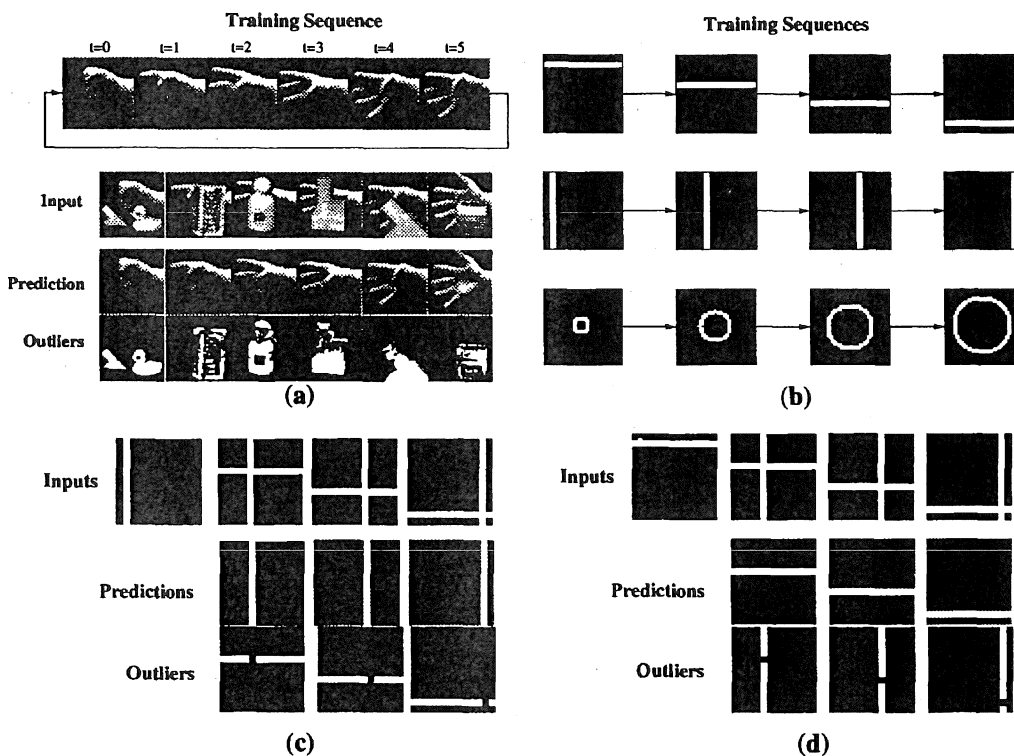

Figure 3: **Correlates of Attention during Dynamic Recognition.** (a) A network was trained on a cyclic image sequence of gestures (top), each image of size $75 \times 75$, with $U$ and $V$ of size $5625 \times 15$ and $15 \times 15$ respectively. The panels below show how the network can ignore various forms of occlusion and clutter (outliers), "attending to" the sequence of gestures that it has been trained on. The outlier threshold $c$ was computed as the mean plus 0.3 standard deviations of the current distribution of squared residual errors. Results shown are those obtained after 5 cycles of exposure to the occluded images. (b) Three image sequences used to train a network. (c) and (d) show the response of the network to ambiguous stimuli comprised of images containing both a horizontal *and* a vertical bar. Note that the network was trained on a horizontal bar moving downwards and a vertical bar moving rightwards (see (b)) but not both simultaneously. Given ambiguous stimuli containing both these stimuli, the network interprets the input differently depending on the initial "priming" input. When the initial input is a vertical bar as in (c), the network interprets the sequence as a vertical bar moving rightwards (with some minor artifacts due to the other training sequences). On the other hand, when the initial input is a horizontal bar as in (d), the sequence is interpreted as a horizontal bar moving downwards, not paying "attention" to the extraneous vertical bars, which are now treated as outliers.

the image region(s) that were previously ignored as outliers. Such a two-step serial recognition process is depicted in Figure 2 (c). The network first recognizes the "dominant" object, which was generally observed to be the object occupying a larger area of the input image or possessing regions with higher contrast. The outlier mask m is subsequently used for "switching attention" and extracting the identity of the second object (lower arrow). Figure 3 shows examples of attention during recognition of dynamic stimuli. In particular, Figure 3 (c) and (d) show how the same image sequence can be interpreted in two different ways depending on which part of the stimulus is "attended to," which in turn depends on the initial priming input.

# 5  CONCLUSIONS

The simulation results indicate that certain experimental observations that have previously been interpreted using the metaphor of an attentional spotlight can also arise as a result of competition and cooperation during visual recognition within networks of linear and non-linear

neurons. Although not explicitly designed to simulate attention, the robust Kalman filter networks nevertheless display some of the essential characteristics of visual attention, such as the preferential processing of a subset of the input signals and the consequent "switching" of attention to previously ignored stimuli. Given multiple objects or conflicting stimuli in their receptive fields (Figures 2 and 3), the responses of the feedforward, feedback, and prediction neurons in the simulated network were modulated according to the current object being "attended to." The modulation in responses was mediated by the non-linear gating neurons $G$, taking into account both bottom-up signals as well top-down feedback signals. This suggests a network-level interpretation of similar forms of attentional response modulation in the primate visual cortex [2, 8, 14], with the consequent prediction that the genesis of attentional modulation in such cases may not necessarily lie within the recorded neurons themselves but within the distributed circuitry that these neurons are an integral part of.

## Footnotes

*This research was supported by NIH/PHS research grant 1-P41-RR09283. I am grateful to Dana Ballard for many useful discussions and suggestions. Author's current address: The Salk Institute, CNL, 10010 N. Torrey Pines Road, La Jolla, CA 92037. E-mail: rao@salk.edu.

[1] Although not implemented here, this "shifting of attentional focus" can be automated using a model of neuronal fatigue and active decay (see, for example, [3]).

# References

[1] A.E. Bryson and Y.-C. Ho. *Applied Optimal Control.* New York: John Wiley, 1975.

[2] L. Chelazzi, E.K. Miller, J. Duncan, and R. Desimone. A neural basis for visual search in inferior temporal cortex. *Nature*, 363:345–347, 1993.

[3] P. Dayan. An hierarchical model of visual rivalry. In M. Mozer, M. Jordan, and T. Petsche, editors, *Advances in Neural Information Processing Systems 9*, pages 48–54. Cambridge, MA: MIT Press, 1997.

[4] A.P. Dempster, N.M. Laird, and D.B. Rubin. Maximum likelihood from incomplete data via the EM algorithm. *J. Royal Statistical Society Series B*, 39:1–38, 1977.

[5] R. Desimone and J. Duncan. Neural mechanisms of selective visual attention. *Annual Review of Neuroscience*, 18:193–222, 1995.

[6] P.J. Huber. *Robust Statistics.* New York: John Wiley, 1981.

[7] R.E. Kalman. A new approach to linear filtering and prediction theory. *Trans. ASME J. Basic Eng.*, 82:35–45, 1960.

[8] J. Moran and R. Desimone. Selective attention gates visual processing in the extrastriate cortex. *Science*, 229:782–784, 1985.

[9] W.T. Newsome. Spotlights, highlights and visual awareness. *Current Biology*, 6(4):357–360, 1996.

[10] E. Niebur and C. Koch. Control of selective visual attention: Modeling the "where" pathway. In D. Touretzky, M. Mozer, and M. Hasselmo, editors, *Advances in Neural Information Processing Systems 8*, pages 802–808. Cambridge, MA: MIT Press, 1996.

[11] B.A. Olshausen, D.C. Van Essen, and C.H. Anderson. A neurobiological model of visual attention and invariant pattern recognition based on dynamic routing of information. *Journal of Neuroscience*, 13:4700–4719, 1993.

[12] R.P.N. Rao and D.H. Ballard. The visual cortex as a hierarchical predictor. Technical Report 96.4, National Resource Laboratory for the Study of Brain and Behavior, Department of Computer Science, University of Rochester, September 1996.

[13] R.P.N. Rao and D.H. Ballard. Dynamic model of visual recognition predicts neural response properties in the visual cortex. *Neural Computation*, 9(4):721–763, 1997.

[14] S. Treue and J.H.R. Maunsell. Attentional modulation of visual motion processing in cortical areas MT and MST. *Nature*, 382:539–541, 1996.

[15] J.K. Tsotsos, S.M. Culhane, W.Y.K. Wai, Y. Lai, N. Davis, and F. Nuflo. Modeling visual attention via selective tuning. *Artificial Intelligence*, 78:507–545, 1995.

